# Diffusion Decision Making for Adaptive $k$-Nearest Neighbor Classification

**Yung-Kyun Noh,   Frank Chongwoo Park**
Schl. of Mechanical and Aerospace Engineering
Seoul National University
Seoul 151-744, Korea
{nohyung,fcp}@snu.ac.kr

**Daniel D. Lee**
Dept. of Electrical and Systems Engineering
University of Pennsylvania
Philadelphia, PA 19104, USA
ddlee@seas.upenn.edu

## Abstract

This paper sheds light on some fundamental connections of the diffusion decision making model of neuroscience and cognitive psychology with $k$-nearest neighbor classification. We show that conventional $k$-nearest neighbor classification can be viewed as a special problem of the diffusion decision model in the asymptotic situation. By applying the optimal strategy associated with the diffusion decision model, an adaptive rule is developed for determining appropriate values of $k$ in $k$-nearest neighbor classification. Making use of the sequential probability ratio test (SPRT) and Bayesian analysis, we propose five different criteria for adaptively acquiring nearest neighbors. Experiments with both synthetic and real datasets demonstrate the effectiveness of our classification criteria.

## 1   Introduction

The recent interest in understanding human perception and behavior from the perspective of neuroscience and cognitive psychology has spurred a revival of interest in mathematical decision theory. One of the standard interpretations of this theory is that when there is a continuous input of noisy information, a decision becomes certain only after accumulating sufficient information. It is also typically understood that early decisions save resources. Among the many theoretical explanations for this phenomenon, the diffusion decision model offers a particularly appealing explanation of how information is accumulated and how the time involved in making a decision affects overall accuracy. The diffusion decision model considers the diffusion of accumulated evidence toward one of the competing choices, and reaches a decision when the evidence meets a pre-defined confidence level.

The diffusion decision model successfully explains the distribution of decision times for humans [13, 14, 15]. More recently, this model offers a compelling explanation of the neuronal decision making process in the lateral intraparietal (LIP) area of the brain for perceptual decision making based on visual evidence [2, 11, 16]. The fundamental premise behind this model is that there is a tradeoff between decision times and accuracy, and that both are controlled by the confidence level. As described in Bogacz et al [3], the sequential probability ratio test (SPRT) is one mathematical model that explains this tradeoff. More recent studies also demonstrate how SPRT can be used to explain the evidence as emanated from Poisson processes [6, 21].

Now shifting our attention to machine learning, the well-known $k$-nearest neighbor classification uses a simple majority voting strategy that, at least in the asymptotic case, implicitly involves a similar tradeoff between time and accuracy. According to Cover and Hart [4], the expected accuracy of $k$-nearest neighbor classification always increases with respect to $k$ when there is sufficient data. At the same time, there is a natural preference to use less resources, or equivalently, a fewer number of nearest neighbors. If one seeks to maximize the accuracy for a given number of total nearest neigh-

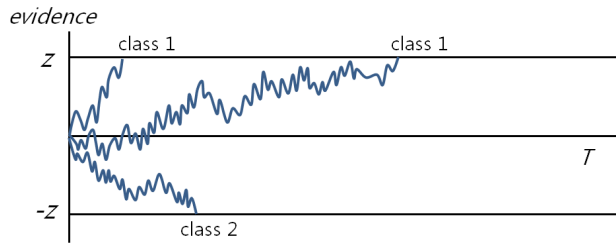

Figure 1: Diffusion decision model. The evidence of decision making is accumulated, and it diffuses over time (to the right). Once the accumulated evidence reaches one of the confidence levels of either choice, $z$ or $-z$, the model stops collecting any more evidence and makes a decision.

bors, this naturally leads to the idea of using different $k$s for different data. At a certain level, this adaptive idea can be anticipated, but methods described in the existing literature are almost exclusively heuristic-based, without offering a thorough understanding of under what situations heuristics are effective [1, 12, 19].

In this work, we present a set of simple, theoretically sound criteria for adaptive $k$-nearest neighbor classification. We first show that the conventional majority voting rule is identical to the diffusion decision model when applied to data from two different Poisson processes. Depending on how the accumulating evidence is defined, it is possible to construct five different criteria based on different statistical tests. First, we derive three different criteria using the SPRT statistical test. Second, using standard Bayesian analysis, we derive two probabilities for the case where one density function is greater than the other. Our five criteria are then used as diffusing evidence; once the evidence exceeds a certain confidence level, collection of information can cease and a decision can be made immediately. Despite the complexity of the derivations involved, the resulting five criteria have a particularly simple and appealing form. This feature can be traced to the memoryless property of Poisson processes. In particular, all criteria can be cast as a function of the information of only one nearest neighbor in each class. Using our derivation, we consider this property to be the result of the assumption that we have sufficient data; the criteria are not guaranteed to work in the event that there is insufficient data. We present experimental results involving real and synthetic data to verify this conjecture.

The remainder of the paper is organized as follows. In Section 2, a particular form of the diffusion decision model is reviewed for Poisson processes, and two simple tests based on SPRT are derived. The relationship between $k$-nearest neighbor classification and diffusion decision making is explained in Section 3. In Section 4, we describe the adaptive $k$-nearest neighbor classification procedure in terms of the diffusion decision model, and we introduce five different criteria within this context. Experiments for synthetic and real datasets are presented in Section 5, and the main conclusions are summarized in Section 6.

## 2 Diffusion Decision Model for Two Poisson Processes

The diffusion decision model is a stochastic model for decision making. The model considers the diffusion of an evidence in favor of either of two possible choices by continuously accumulating the information. After initial wavering between the two choices, the evidence finally reaches a level of confidence where a decision is made as in Fig. 1.

In mathematical modeling of this diffusion process, Gaussian noise has been predominantly used as a model for zigzagging upon a constant drift toward a choice [3, 13]. However, when we consider two competing Poisson signals, a simpler statistical test can be used instead of estimating the direction of the drift. In the studies of decision making in the lateral intraparietal (LIP) area of the brain [2, 11], two Poisson processes are assumed to have rate parameters of either $\lambda_+$ and $\lambda_-$ where we know that $\lambda_+ > \lambda_-$, but exact values are unknown. When it should be determined which Poisson process has the larger rate $\lambda_+$, a sequential probability ratio test (SPRT) can be used to explain a diffusion decision model [6, 21].

The Poisson distribution we use has the form: $p(N|\lambda, T) = \frac{(\lambda T)^N}{N!} \exp(-\lambda T)$, and we consider two Poisson distributions for $N_1$ and $N_2$ at time $T_1$ and $T_2$, respectively: $p(N_1|\lambda_1, T_1)$ and $p(N_2|\lambda_2, T_2)$. Here, $\lambda_1$ and $\lambda_2$ are the rate parameters, and either of these parameters has $\lambda_+$ where the other has $\lambda_-$. Now, we apply the statistical test of Wald [18] for a confidence $\alpha(>1)$:

$$\frac{p(N_1|\lambda_1 = \lambda_+)p(N_2|\lambda_2 = \lambda_-)}{p(N_1|\lambda_1 = \lambda_-)p(N_2|\lambda_2 = \lambda_+)} > \alpha \quad \text{or} \quad < \frac{1}{\alpha} \tag{1}$$

for the situation where there is $N_1$ number of signals at time $T_1$ for the first Poisson process and $N_2$ number of signals at time $T_2$ for the second process. We can determine that $\lambda_1$ has the $\lambda_+$ once the left term is greater than $\alpha$, and $\lambda_2$ has the $\lambda_+$ once it is greater than $\frac{1}{\alpha}$, otherwise, we must collect more information. According to Wald and Wolfowitz [18], this test is optimal in that the test requires the fewest average observations with the same probability of error.

By taking the log on both sides, we can rewrite the test as

$$\log\left(\frac{\lambda_+}{\lambda_-}\right)(N_1 - N_2) - (\lambda_+ - \lambda_-)(T_1 - T_2) \quad > \quad \log a \qquad \text{or} \tag{2}$$
$$< -\log a.$$

Considering two special situations, this equation can be reduced into two different, simple tests. First, we can consider observation of the numbers $N_1$ and $N_2$ at a certain time $T = T_1 = T_2$. Then test in Eq. (2) is reduced into one test previously proposed in [21]:

$$|N_1 - N_2| > z_N \tag{3}$$

where $z_N$ is a constant satisfying $z_N = \frac{\log \alpha}{\log(\lambda_+/\lambda_-)}$. Another simple test can be made by using the observation times $T_1$ and $T_2$ when we find the same number of signals $N = N_1 = N_2$:

$$|T_1 - T_2| > z_T \tag{4}$$

where $z_T$ satisfies $z_T = \frac{\log \alpha}{\lambda_+ - \lambda_-}$.

Here, we can consider $\Delta N = N_1 - N_2$ and $\Delta T = T_1 - T_2$ as two different evidences in the diffusion decision model. The evidence diffuses as we collect more information, and we come to make a decision once the evidence reaches the confidence levels, $\pm z_N$ for $\Delta N$, and $\pm z_T$ for $\Delta T$. In this work, we refer to the first model, using the criterion $\Delta N$, as the $\Delta N$ rule and the second model, using $\Delta T$, as the $\Delta T$ rule.

Although the $\Delta N$ rule has been previously derived and used [21], we propse four more test criteria in this paper including Eq. (4). Later, we show that the diffusion decision making with these five criteria is related to different methods for $k$-nearest neighbor classification.

## 3 Equivalence of Diffusion Decision Model and $k$-Nearest Neighbor Classification

A conventional $k$-nearest neighbor ($k$-NN) classification takes a majority voting strategy using $k$ number of nearest neighbors. According to Cover and Hart [4], in the limit of infinite sampling, this simple majority voting rule can produce a fairly low expected error and furthermore, this error decreases even more as a bigger $k$ is used. This theoretical result is obtained from the relationship between the $k$-NN classification error and the optimal Bayes error: the expected error with one nearest neighbor is always less than twice the Bayes error, and the error decreases with the number of $k$ asymptotically to the Bayes error [4].

In this situation, we can claim that the $k$-NN classification actually performs the aforementioned diffusion decision making for Poisson processes. The identity comes from two equivalence relationships: first, the logical equivalence between two decision rules; second, the equivalence of distribution of nearest neighbors to the Poisson distribution in an asymptotic situation.

### 3.1 Equivalent Strategy of Majority Voting

Here, we first show an equivalence between the conventional $k$-NN classification and a novel comparison algorithm:

**Theorem:** *For two-class data, we consider the $N$-th nearest datum of each class from the testing point. With an odd number $k$, majority voting rule in $k$-NN classification is equivalent to the rule of picking up the class to which a datum with smaller distance to the testing point belongs, for $k = 2N - 1$.*

**Proof:** Among $k$-NNs of a test point, if there are more than or equal to $N$ data having label $C$, for $C \in \{1, 2\}$, the test point is classified as class $C$ according to the majority voting because $N = (k+1)/2 > \frac{k}{2}$. If we consider three distances $d_k$ to the $k$-th nearest neighbor among all data, $d_{N,C}$ to the $N$-th nearest neighbor in class $C$, and $d_{N,\neg C}$ to the $N$-th nearest neighbor in class non-$C$, then both $d_{N,C} \leq d_k$ and $d_{N,\neg C} > d_k$ are satisfied in this case. This completes one direction of proof that the selection of class $C$ by majority voting implies $d_{N,C} < d_{N,\neg C}$. The opposite direction can be proved similarly. ∎

Therefore, instead of counting the number of nearest neighbors, we can classify a test point using two separate $N$-th nearest neighbors of two classes and comparing the distances. This logical equivalence applies regardless of the underlying density functions.

### 3.2   Nearest neighbors as Poisson processes

The random generation of data from a particular underlying density function induces a density function of distance to the nearest neighbors. When the density function is $\lambda(\mathbf{x})$ for $\mathbf{x} \in \mathbb{R}^D$ and we consider a $D$-dimensional hypersphere of volume $V$ with $N$-th nearest neighbor on its surface, a random variable $u = MV$, which is the volume of the sphere $V$ multiplied by the number of data $M$, asymptotically converges in distribution to the Erlang density function [10]:

$$p(u|\lambda) = \frac{\lambda^N}{\Gamma(N)} \exp(-\lambda u) u^{N-1} \tag{5}$$

with a large amount of data. Here, the volume element is a function of distance $d$ which can be represented as $V = \gamma d^D$ and $\gamma = \frac{\pi^{D/2}}{\Gamma(D/2+1)}$, a proportionality constant for a hypersphere volume. This Erlang function is a special case of the Gamma density function when the parameter $N$ is an integer.

We can also note that this Erlang density function implies the Poisson distribution with respect to $N$ [20], and we can write the distribution of $N$ as follows:

$$p(N|\lambda) = \frac{\lambda^N}{\Gamma(N+1)} \exp(-\lambda). \tag{6}$$

This equation shows that the appearance of nearest neighbors can be approximated with Poisson processes. In other words, with a growing hypersphere at a constant rate in volume, the occurrence of new points within a hypersphere will follow a Poisson distribution.

This Erlang function in Eq. (5) comes from the asymptotic convergence in distribution of the real distribution, the binomial distribution with finite $N$ number of samples [10]. Here, we note that, with a finite number of samples, the memoryless property of the Poisson disappears. This results in the breakdown of the independency assumption between posterior probabilities for classes which Cover and Hart used implicitly when they derived the expected error of $k$-NN classification [4].

On the other hand, once we have enough data, and hence the density functions Eq. (5) and Eq. (6) explain data correctly, we can expect the equivalence between the diffusion decision making and $k$-NN classification. In this case, the nearest neighbors are the samples of a Poisson process, having the rate parameter $\lambda$, which is the probability density at the test point.

Now, we can turn back to the conventional $k$-NN classification. By theorem 1 and the arguments in this section, the $k$-NN classification strategy is the same as the strategy of comparing two Poisson processes using $N$-th samples of each class. This connection naturally exploits the conventional $k$-NN classification to the adaptive method of using different $k$s using the confidence level in the diffusion decision model.

# 4 Criteria for Adaptive $k$-NN Classification

Using the equivalence settings of the diffusion decision model and the $k$-NN classification, we can extend the conventional majority voting strategy to more sophisticated adaptive strategies. First, the SPRT criteria in the previous section, $\Delta N$ rule and $\Delta T$ rule can be used. For the $\Delta N$ rule in Eq. (3), we can use the numbers of nearest neighbors $N_1$ and $N_2$ within a fixed distance $d$, then compare $|\Delta N| = |N_1 - N_2|$ with a pre-defined confidence level $z_N$. Instead of making an immediate decision, we can collect more nearst neighbors by increasing $d$ until Eq. (3) is satisfied. This is the "$\Delta N$ rule" for adaptive $k$-NN classification.

In terms of the $\Delta T$ rule in Eq. (4), using the correspondence of time in the original SPRT to the volume within the hypersphere in $k$-NN classification, we can make two different criteria for adaptive $k$-NN classification. First, we consider two volume elements, $V_1$ and $V_2$ of $N$-th nearest neighbors, and the criterion can be rewritten as $|V_1 - V_2| > z_V$. We refer to this rule as the "$\Delta V$ rule".

Additional criterion for the $\Delta T$ rule considers a more conservative rule using the volume of $(N+1)$-th nearest neighbor hypersphere. Since a slightly smaller hypersphere than this hypersphere still contains $N$ number of nearest neighbors, we can make the *same* test more difficult to stop diffusing by replacing the smaller volume in the $\Delta V$ rule with the volume of $(N + 1)$-th nearest neighbor hypersphere of that class. We refer to this rule as the "Conservative $\Delta V$ rule" because it is more cautious in making a decision with this strategy.

In addition to the SPRT method, with which we derive three different criteria, we can also derive several stopping criteria using the Bayesian approach. If we consider $\lambda$ as a random variable and apply an appropriate prior, we can obtain a posterior distribution of $\lambda$ as well as the probability of $P(\lambda_1 > \lambda_2)$ or $P(\lambda_1 < \lambda_2)$. In the following section, we show how we can derive these probabilities and how these probabilities can be used as evidence in the diffusion decision making model.

## 4.1 Bayesian Criteria

For both Eq. (5) and Eq. (6), we consider $\lambda$ as a random variable, and we can apply a conjugate prior for $\lambda$:

$$p(\lambda) = \frac{b^a}{\Gamma(a)} \lambda^{a-1} \exp(-\lambda b) \tag{7}$$

with constants $a$ and $b$. The constant $a$ is an integer satisfying $a \geq 1$, and $b$ is a real number. With this prior Eq. (7), the posteriors for two likelihoods Eq. (5) and Eq. (6) are obtained easily:

$$p(\lambda|u) = \frac{(u+b)^{N+a}}{\Gamma(N+a)} \lambda^{N+a-1} \exp(-\lambda(u+b)) \tag{8}$$

$$p(\lambda|N) = \frac{(b+1)^{N+a}}{\Gamma(N+a)} \lambda^{N+a-1} \exp(-\lambda(b+1)) \tag{9}$$

First, we derive $P(\lambda_1 > \lambda_2|u_1, u_2)$ for $u_1$ and $u_2$ obtained using the $N$-th nearest neighbors in class 1 and class 2. Because the posterior functions of different classes are independent from each other, this probability of $\lambda_1 > \lambda_2$ is simply obtained by the double integration:

$$P(\lambda_1 > \lambda_2|u_1, u_2) = \int_0^\infty p(\lambda_2|u_2) \int_0^{\lambda_2} p(\lambda_1|u_1) \, d\lambda_1 \, d\lambda_2. \tag{10}$$

After some calculation, the integration result gives an extremely simple analytic solution:

$$P(\lambda_1 > \lambda_2|u_1, u_2) = \sum_{m=0}^{N+a-1} \binom{2N+2a-1}{m} \frac{(u_1+b)^m(u_2+b)^{2N+2a-1-m}}{(u_1+u_2+2b)^{2N+2a-1}} \tag{11}$$

Here, we merely consider the case that $a = 1$, and it is interesting to note that this probability is equivalent to the probability of flipping a biased coin $2N + 1$ times and observing less than or equal to $N$ number of heads. This probability from the Bayesian approach can be efficiently computed

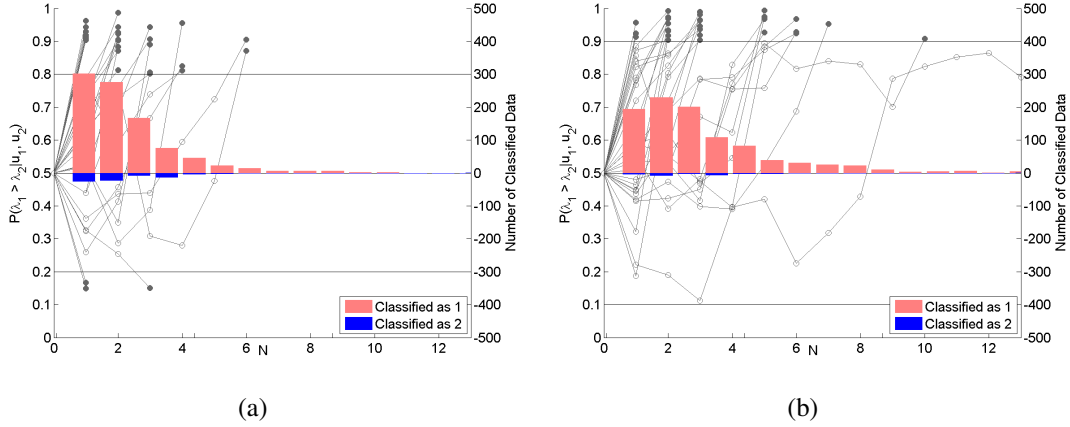

<div align="center">(a)</div>

<div align="center">(b)</div>

Figure 2: Decision making process for the nearest neighbor classification with (a) 80% and (b) 90% confidence level. Sample data are generated from the probability densities $\lambda_1 = 0.8$ and $\lambda_2 = 0.2$. For incrementing $N$-th nearest neighbors of different classes, the criterion probabilities $P(\lambda_1 > \lambda_2 | u_1, u_2)$ and $P(\lambda_1 < \lambda_2 | u_1, u_2)$ are calculated and compared with the confidence level. Unless the probability exceeds the confidence level, the next $(N + 1)$-th nearest neighbors are collected and the criterion probabilities are calculated again. In this figure, the diffusion of the criterion probability $P(\lambda_1 > \lambda_2 | u_1, u_2)$ is displayed for different realizations, where the evidence stops diffusing once the criterion passes the threshold where enough evidence has accumulated. The bars represent the number of points that are correctly (Red, upward bars) and incorrectly (Blue, downward bars) classified at each stage of the computation. Using a larger confidence results in less error, but with a concomitant increase in the number of nearest neighbors used.

in an incremental fashion, and the nearest neighbor computation can be adaptively stopped with enough confidence of the evidence probability.

The second probability $P(\lambda_1 > \lambda_2 | N_1, N_2)$ for the number of nearest neighbors $N_1$ and $N_2$ within a particular distance can be similarly derived. Using the double integration of Eq. (9), we can derive the analytic result again as

$$P(\lambda_1 > \lambda_2 | N_1, N_2) = \frac{1}{2^{N_1 + N_2 + 2a - 1}} \sum_{m=0}^{N_1 + a - 1} \left( \begin{array}{c} N_1 + N_2 + 2a - 1 \\ m \end{array} \right). \tag{12}$$

Both the probabilities Eq. (11) and Eq. (12) can be used as evidence that diffuse along with incoming information. Stopping criteria for diffusion can be derived using these probabilities.

## 4.2 Adaptive $k$-NN Classification

Of interest in the diffusion decision model is the relationship between the accuracy and the amount of resources needed to obtain the accuracy. In a diffusion decision setting for $k$-NN classification, we can control the amount of resources using the confidence level. For example, in Fig. 2, we generated data from two uniform density functions, $\lambda_1 = 0.8$ and $\lambda_2 = 0.2$, for different classes, and we applied different confidence levels, 0.8 and 0.9 in Fig. 2(a) and (b), respectively. Using the $P(\lambda_1 > \lambda_2 | u_1, u_2)$ criterion in Eq. (11), we applied the adaptive $k$-NN classification with an increasing $N$ of two classes.

Fig. 2 shows the decision results of the classification with incrementing $N$ for 1000 realizations, and a few diffusion examples of the evidence probability in Eq. (11) are presented. According to the confidence level, the average number of nearest neighbors used differs. For Fig. 2(a) when the confidence level is lower than Fig. 2(b), the evidence reaches the confidence level at an earlier stage than Fig. 2(b), while the decision in Fig. 2(b) tends to select the first class more often than in Fig. 2(a). Considering that the optimal Bayes classification choosing class 1 for $\lambda_1 > \lambda_2$, the decisions for class 2 can be considered as errors. In this sense, we can say with the higher confidence level, decisions are made more correctly while using more resources. Therefore, the efficiencies

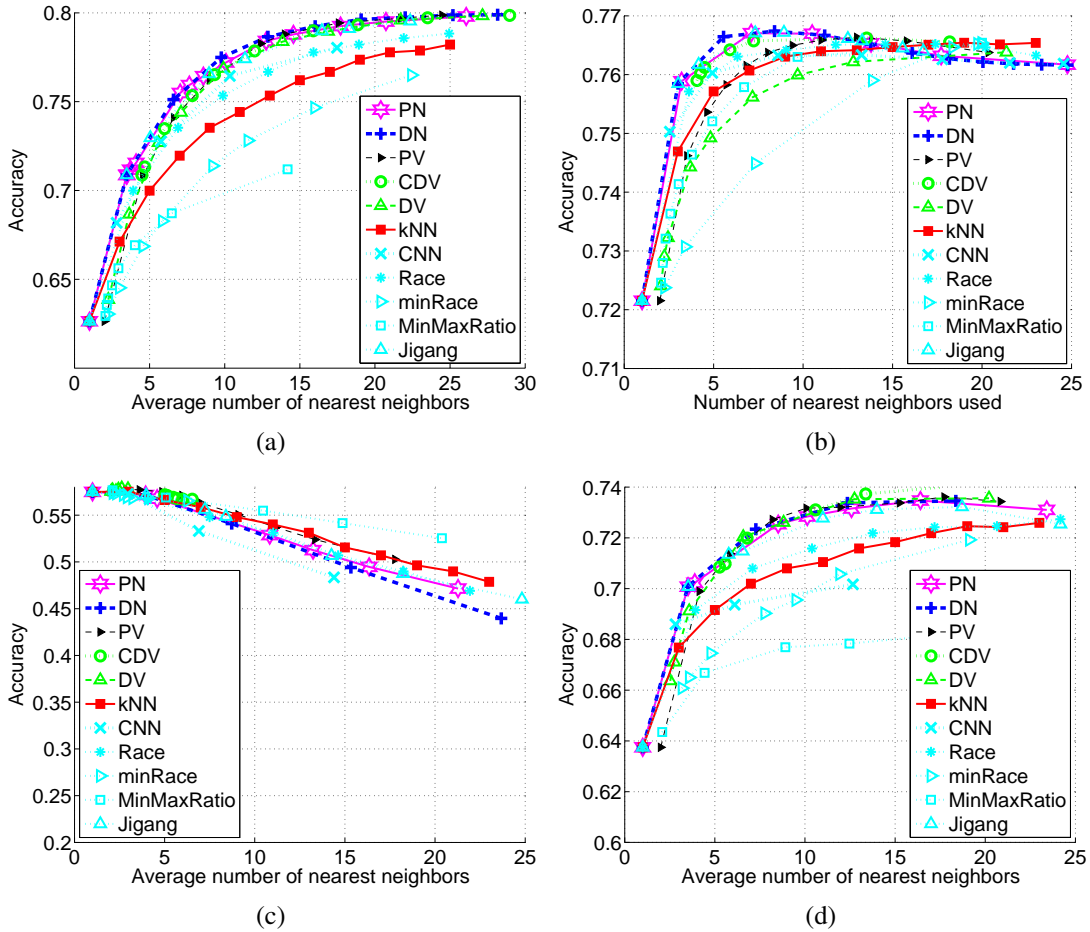

Figure 3: Classification accuracy (vertical axis) versus the average number of nearest neighbors used (horizontal axis) for adaptive $k$-NN classification. (a) Uniform probability densities for $\lambda_1 = 0.8$ and $\lambda_2 = 0.2$ in 100-dimensional space, (b) CIFAR-10, (c) $2 \times 10^5$ data per class for 5-dimensional Gaussians, and (d) $2 \times 10^6$ data per class for the same Gaussians in (c) are used.

between strategies can be compared using the accuracies as well as the average number of nearest neighbors used.

## 5 Experiments

In the experiments, we compare the accuracy of the algorithms to the number of nearest neighbors used, for various confidence levels for criteria. We used the conventional $k$-NN classification as well as the proposed adaptive methods. Adaptive classification includes the comparison rule of $N$-th nearest neighbors using three criteria—the $\Delta V$ rule (DV), the Conservative $\Delta V$ rule (CDV), and Bayesian probability in Eq. (11) (PV)—, as well as the comparison rule of $N_1$-th and $N_2$-th at a given volume using two rules—the $\Delta N$ rule (DN) and Bayesian probability in Eq. (12) (PN). We present the average accuracies resulting from the use of these $k$-NN classification and five adaptive rules with respect to the average number of nearest neighbors used.

We first show the results on synthetic datasets. In Fig. 3(a), we used two uniform probability densities $\lambda_1 = 0.8$ and $\lambda_2 = 0.2$ in 100-dimensional space, and we classified a test point based on the nearest neighbors. In this figure, all algorithms are expected to approach the Bayes performance based on Cover and Hart's approach when the average number of nearest neighbors increase. In

this experiment, we can observe that all five proposed adaptive algorithms approach the Bayes error quicker than other methods showing similar rates with each other.

Here, we also present the results of other adaptive algorithms CNN [12], Race, minRace, Min-MaxRatio, and Jigang [19]. They perform majority voting with increasing $k$; CNN stops collecting more nearest neighbors once more than a certain amount of consecutive neighbors are found with the same labels; Race stops when the total amount of neighbors of one class exceeds a certain level; minRace stops when all classes have at least a predefined amount of neighbors; MinMaxRatio considers the ratio between numbers of nearest neighbors in different classes; lastly, Jigang is a probability criterion slightly different from Eq. (12). Except for Jigang's method, all algorithms perform poorly, while our five algorithms perform equally well though they use different information, probably because the performance produced by diffusion decision making algorithms is optimal.

Fig. 3(b) shows the experiments for a CIFAR-10 subset of the tiny images dataset [17]. The CIFAR-10 set has 10-class $32 \times 32$ color images. Each class has 6000 images, and they are separated into one testing set and five training sets. With this 10-class data, we first performed Fisher Discriminant Analysis to obtain a 9-dimensional subspace, then all different adaptive algorithms are applied on this subspace. The result is the average accuracy for five different training sets and for all possible pairs of 10 classes. Because the underlying density is non-uniform here, the result shows the performance decrease when algorithms use non-close nearest neighbors. Except for DV and PV criteria, all of our adaptive algorithms outperform all other methods. The $k$-NN classification in the original data space shows the maximal average performance of $0.721$ at $k = 3$, which is far less than the overall accuracies in the figure, because the distance information is poor in the high dimensional space.

Fig. 3(c) and (d) clearly show that our algorithms are *not* guaranteed to work with insufficient data. We generated data from two different Gaussian functions and tried to classify a datum located at one of the modes to figure out the label of this datum. The number of generated data is $2 \times 10^5$ per class for (c), and $2 \times 10^6$ per class for (d) in 5-dimensional space. We presented the average result of 5000 realizations, and the comparison of two figures show that our adaptive algorithms work as expected when Cover and Hart's asymptotic data condition holds. The Poisson process assumption also holds when this condition is satisfied.

# 6 Conclusions

In this work, we showed that $k$-NN classification in the asymptotic limit is equivalent to the diffusion decision model for decision making. Nearest neighbor classification and the diffusion decision model are both very well known models in machine learning and cognitive science respectively, but the intimate connection between them has not been studied before. Using analysis of Poisson processes, we showed how classification using incrementally increasing nearest neighbors can be mapped to a simple threshold based decision model.

In the diffusion decision model, the confidence level plays a key role in determining the tradeoff between speed and accuracy. The notion of confidence can also be applied to nearest neighbor classification to adapt the number of nearest neighbors used in making the classification decision. We presented several different criteria for choosing the appropriate number of nearest neighbors based on the sequential probability ratio test in addition to Bayesian inference. We demonstrated the utility of these methods in modulating speed versus accuracy on both simulated and benchmark datasets.

It is straightforward to extend these methods to other datasets and algorithms that utilize neighborhood information. Future work will investigate how our results would scale with dataset size and feature representations. Potential benefits of this work include a well-grounded approach to speeding up classification using parallel computation on very large datasets.

**Acknowledgments**

This research is supported in part by the US Office of Naval Research, Intel Science and Technology Center, AIM Center, KIST-CIR, ROSAEC-ERC, SNU-IAMD, and the BK21.

# References

[1] A. F. Atiya. Estimating the posterior probabilities using the $k$-nearest neighbor rule. *Neural Computation*, 17(3):731–740, 2005.

[2] J. M. Beck, W. J. Ma, R. Kiani, T. Hanks, A. K. Churchland, J. Roitman, M. N. Shadlen, P. E. Latham, and A. Pouget. Probabilistic population codes for Bayesian decision making. *Neuron*, 60(6):1142–1152, 2008.

[3] R. Bogacz, E. Brown, J. Moehlis, P. Holmes, and J. D. Cohen. The physics of optimal decision making: A formal analysis of models of performance in two-alternative forced-choice tasks. *Psychological Review*, 113(4):700–765, 2006.

[4] T. Cover and P. Hart. Nearest neighbor pattern classification. *IEEE Transactions on Information Theory*, 13(1):21–27, 1967.

[5] L. Devroye, L. Györfi, and G. Lugosi. *A probabilistic theory of pattern recognition*. Applications of mathematics. Springer, 1996.

[6] M. A. Girshick. Contributions to the theory of sequential analysis I. *The Annuals of Mathematical Statistics*, 17:123–143, 1946.

[7] M. Goldstein. $k_n$-Nearest Neighbor Classification. *IEEE Transactions on Information Theory*, IT-18(5):627–630, 1972.

[8] C. C. Holmes and N. M. Adams. A probabilistic nearest neighbour method for statistical pattern recognition. *Journal of the Royal Statistical Society Series B*, 64(2):295–306, 2002.

[9] M. D. Lee, I. G. Fuss, and D. J. Navarro. A Bayesian approach to diffusion models of decision-making and response time. In *Advances in Neural Information Processing Systems 19*, pages 809–816. 2007.

[10] N. Leonenko, L. Pronzato, and V. Savani. A class of Rényi information estimators for multidimensional densities. *Annals of Statistics*, 36:2153–2182, 2008.

[11] W. J. Ma, J. M. Beck, P. E. Latham, and A. Pouget. Bayesian inference with probabilistic population codes. *Nature Neuroscience*, 9(11):1432–1438, 2006.

[12] S. Ougiaroglou, A. Nanopoulos, A. N. Papadopoulos, Y. Manolopoulos, and T. Welzer-Druzovec. Adaptive k-nearest-neighbor classification using a dynamic number of nearest neighbors. In *Proceedings of the 11th East European conference on Advances in databases and information systems*, pages 66–82, 2007.

[13] R. Ratcliff and G. Mckoon. The diffusion decision model: theory and data for two-choice decision tasks. *Neural Computation*, 20(4):873–922, 2008.

[14] R. Ratcliff and J. N. Rouder. A diffusion model account of masking in two-choice letter identification. *Journal of Experimental Psychology Human Perception and Performance*, 26(1):127–140, 2000.

[15] M. N. Shadlen, A. K. Hanks, A. K. Churchland, R. Kiani, and T. Yang. The speed and accuracy of a simple perceptual decision: a mathematical primer. *Bayesian brain: Probabilistic approaches to neural coding*, 2006.

[16] M. N. Shadlen and W. T. Newsome. The variable discharge of cortical neurons: Implications for connectivity, computation, and information coding. *Journal of Neuroscience*, 18:3870–3896, 1998.

[17] A. Torralba, R. Fergus, and W. T. Freeman. 80 million tiny images: A large data set for non-parametric object and scene recognition. *IEEE Transactions on Pattern Analysis and Machine Intelligence*, 30(11):1958–1970, 2008.

[18] A. Wald and J. Wolfowitz. Optimum character of the sequential probability ratio test. *Annals of Mathematical Statistics*, 19:326–339, 1948.

[19] J. Wang, P. Neskovic, and L. N. Cooper. Neighborhood size selection in the k-nearest-neighbor rule using statistical confidence. *Pattern Recognition*, 39(3):417–423, 2006.

[20] L. Wasserman. *All of Statistics: A Concise Course in Statistical Inference (Springer Texts in Statistics)*. Springer, December 2003.

[21] J. Zhang and R. Bogacz. Optimal decision making on the basis of evidence represented in spike trains. *Neural Computation*, 22(5):1113–1148, 2010.

